# The Connectivity Analysis of Simple Association
## - or -
## How Many Connections Do You Need?

*Dan Hammerstrom**
Oregon Graduate Center, Beaverton, OR 97006

## ABSTRACT

The efficient realization, using current silicon technology, of Very Large Connection Networks (VLCN) with more than a billion connections requires that these networks exhibit a high degree of communication locality. Real neural networks exhibit significant locality, yet most connectionist/neural network models have little. In this paper, the connectivity requirements of a simple associative network are analyzed using communication theory. Several techniques based on communication theory are presented that improve the robustness of the network in the face of sparse, local interconnect structures. Also discussed are some potential problems when information is distributed too widely.

## INTRODUCTION

Connectionist/neural network researchers are learning to program networks that exhibit a broad range of cognitive behavior. Unfortunately, existing computer systems are limited in their ability to emulate such networks efficiently. The cost of emulating a network, whether with special purpose, highly parallel, silicon-based architectures, or with traditional parallel architectures, is *directly* proportional to the number of connections in the network. This number tends to increase geometrically as the number of nodes increases. Even with large, massively parallel architectures, connections take time and silicon area. Many existing neural network models scale poorly in learning time and connections, precluding large implementations.

The connectivity costs of a network are directly related to its locality. A network exhibits *locality of communication* [1] if most of its processing elements connect to other physically adjacent processing elements in any reasonable mapping of the elements onto a planar surface. There is much evidence that real neural networks exhibit locality[2]. In this paper, a technique is presented for analyzing the effects of locality on the process of association. These networks use a complex node similar to the higher-order learning units of Maxwell et al. [3].

## NETWORK MODEL

The network model used in this paper is now defined (see Figure 1).

*Definition 1:* A *recursive neural network*, called a *c-graph* is a graph structure, $\Gamma(V,E,C)$, where:

- There is a set of CNs (network nodes), $V$, whose outputs can take a range of positive real values, $v_i$, between 0 and 1. There are $N_v$ nodes in the set.

- There is a set of *codons*, $E$, that can take a range of positive real values, $e_{ij}$ (for codon $j$ of node $i$), between 0 and 1. There are $N_c$ codons dedicated to each CN (the output of each codon is only used by its local CN), so there are a total of $N_c N_v$ codons in the network. The fan-in or *order* of a codon is $f_c$. It is assumed that $f_c$ is the same for each codon, and $N_c$ is the same for each CN.

*This work was supported in part by the Semiconductor Research Corporation contract no. 86-10-097, and jointly by the Office of Naval Research and Air Force Office of Scientific Research, ONR contract no. N00014 87 K 0259.

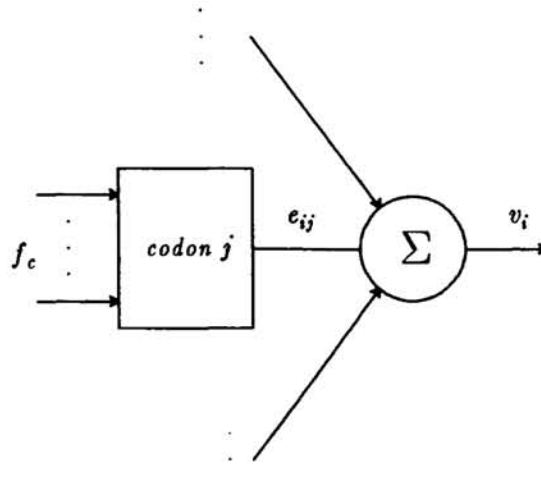

Figure 1 - A CN

- $c_{ijk} \in C$ is a set of *connections* of CNs to codons, $1 \leq i,k \leq N_v$ and $1 \leq j \leq N_c$, $c_{ijk}$ can take two values $\{0,1\}$ indicating the existence of a connection from CN $k$ to codon $j$ of CN $i$. □

*Definition 2:* The value of CN $i$ is

$$v_i = F\left[\theta + \sum_{j=1}^{N_c} e_{ij}\right] \tag{1}$$

The function, $F$, is a continuous non-linear, monotonic function, such as the sigmoid function. □

*Definition 3:* Define a mapping, $D(i,j,\vec{x}) \to \vec{y}$, where $\vec{x}$ is an input vector to $\Gamma$ and $\vec{y}$ is the $f_c$ element input vector of codon $j$ of CN $i$. That is, $\vec{y}$ has as its elements those elements of $x_k$ of $\vec{x}$ where $c_{ijk} = 1$, $\forall\ k$. □

The $D$ function indicates the subset of $\vec{x}$ seen by codon $j$ of CN $i$. Different input vectors may map to the same codon vectors, e.g., $D(i,j,\vec{x}) \to \vec{y}$ and $D(i,j,\vec{z}) \to \vec{y}$, where $\vec{x} \neq \vec{z}$.

*Definition 4:* The codon values $e_{ij}$ are determined as follows. Let $\vec{x}(m)$ be input vector $m$ of the $M$ learned input vectors for CN $i$. For codon $e_{ij}$ of CN $i$, let $T_{ij}$ be the set of $f_c$-dimensional vectors such that $\vec{t}_{ij}(m) \in T_{ij}$, and $D(i,j,\vec{x}(m)) \to \vec{t}_{ij}(m)$. That is, each vector, $\vec{t}_{ij}(m)$ in $T_{ij}$ consists of those subvectors of $\vec{x}(m)$ that are in codon $ij$'s receptive field.

The variable $l$ indexes the $L(i,j)$ vectors of $T_{ij}$. The number of distinct vectors in $T_{ij}$ may be less than the total number of learned vectors $(L(i,j) \leq M)$. Though the $\vec{x}(m)$ are distinct, the subsets, $\vec{t}_{ij}(m)$, need not be, since there is a possible many to one mapping of the $\vec{x}$ vectors onto each vector $\vec{t}_{ij}$.

Let $X^1$ be the subset of vectors where $v_i = 1$ (CN $i$ is supposed to output a 1), and $X^0$ be those vectors where $v_i = 0$, then define

$$n_{ij}^q(l) = size\_of\left\{D(i,j,\vec{x}(m))\ st\ v_i = q\right\} \tag{2}$$

for $q = 0,1$, and $\forall\ m$ that map to this $l$. That is, $n_{ij}^0(l)$ is the number of $\vec{x}$ vectors that map

into $T_{ij}^{\cdot}(l)$ where $v_i=0$ and $n_{ij}^1(l)$ is the number of $\vec{x}$ vectors that map into $T_{ij}^{\cdot}(l)$, where $v_i=1$.

The *compression* of a codon for a vector $T_{ij}^{\cdot}(l)$ then is defined as

$$HC_{ij}(l) = \frac{n_{ij}^1(l)}{n_{ij}^1(l)+n_{ij}^0(l)} \tag{3}$$

($HC_{ij}(l)\equiv 0$ when both $n^1$, $n^0=0$.) The output of codon $ij$, $e_{ij}$, is the maximum-likelyhood decoding

$$e_{ij} = HC_{ij}(l'). \tag{4}$$

Where $HC$ indicates the likelyhood of $v_i=1$ when a vector $\vec{x}$ that maps to $l'$ is input, and $l'$ is that vector $T(l')$ where $min[d_h(T(l'),\vec{y})]$ $\forall$ $l$, $D(i,j,\vec{x}) \to \vec{y}$, and $\vec{x}$ is the current input vector. In other words, $l'$ is that vector (of the set of subset learned vectors that codon $ij$ receives) that is closest (using distance measure $d_h$) to $\vec{y}$ (the subset input vector). □

The output of a codon is the "most-likely" output according to its inputs. For example, when there is no code compression at a codon, $e_{ij}=1$, if the "closest" (in terms of some measure of vector distance, e.g. Hamming distance) subvector in the receptive field of the codon belongs to a learned vector where the CN is to output a 1. The codons described here are very similar to those proposed by Marr [4] and implement nearest-neighbor classification. It is assumed that codon function is determined statically prior to network operation, that is, the desired categories have already been learned.

To measure performance, network capacity is used.

*Definition 5:* The *input noise*, $\Omega_I$, is the average $d_h$ between an input vector and the closest (minimum $d_h$) learned vector, where $d_h$ is a measure of the "difference" between two vectors - for bit vectors this can be Hamming distance. The *output noise*, $\Omega_O$, is the average distance between network output and the learned output vector associated with the closest learned input vector. The *information gain*, $G_I$, is just

$$G_I \equiv -\log\left(\frac{\Omega_I}{\Omega_O}\right) \tag{5}$$

□

*Definition 6:* The *capacity* of a network is the maximum number of learned vectors such that the information gain, $G_I$, is strictly positive (>0). □

## COMMUNICATION ANALOGY

Consider a single connection network node, or CN. (The remainder of this paper will be restricted to a single CN.) Assume that the CN output value space is restricted to two values, 0 and 1. Therefore, the CN must decide whether the input it sees belongs to the class of "0" codes, those codes for which it remains off, or the class of "1" codes, those codes for which it becomes active. The inputs it sees in its receptive field constitute a subset of the input vectors (the $D(...)$ function) to the network. It is also assumed that the CN is an *ideal* 1-NN (Nearest Neighbor) classifier or feature detector. That is, given a particular set of learned vectors, the CN will classify an arbitrary input according to the class of the nearest (using $d_h$ as a measure of distance) learned vector. This situation is equivalent to the case where a single CN has a single codon whose receptive field size is equivalent to that of the CN.

Imagine a sender who wishes to send one bit of information over a noisy channel. The sender has a probabilistic encoder that choses a code word (learned vector) according to some probability distribution. The receiver knows this code set, though it has no knowledge of which bit is being sent. Noise is added to the code word during its transmission over the

channel, which is analogous to applying an input vector to a network's inputs, where the vector lies within some learned vector's region. The "noise" is represented by the distance $(d_k)$ between the input vector and the associated learned vector.

The code word sent over the channel consists of those bits that are seen in the receptive field of the CN being modeled. In the associative mapping of input vectors to output vectors, each CN must respond with the appropriate output (0 or 1) for the associated learned output vector. Therefore, a CN is a decoder that estimates in which class the received code word belongs. This is a classic block encoding problem, where increasing the field size is equivalent to increasing code length. As the receptive field size increases, the performance of the decoder improves in the presence of noise. Using communication theory then, the trade-off between interconnection costs as they relate to field size and the functionality of a node as it relates to the correctness of its decision making process (output errors) can be characterized.

As the receptive field size of a node increases, so does the redundancy of the input, though this is dependent on the particular codes being used for the learned vectors, since there are situations where increasing the field size provides no additional information. There is a point of diminishing returns, where each additional bit provides ever less reduction in output error. Another factor is that interconnection costs increase exponentially with field size. The result of these two trends is a cost performance measure that has a single global maximum value. In other words, given a set of learned vectors and their probabilities, and a set of interconnection costs, a "best" receptive field size can be determined, beyond which, increasing connectivity brings diminishing returns.

## SINGLE CODON, WITH NO CODE COMPRESSION

A single neural element with a single codon and with no code compression can be modelled exactly as a communication channel (see Figure 2). Each network node is assumed to have a single codon whose receptive field size is equal to that of the receptive field size of the node.

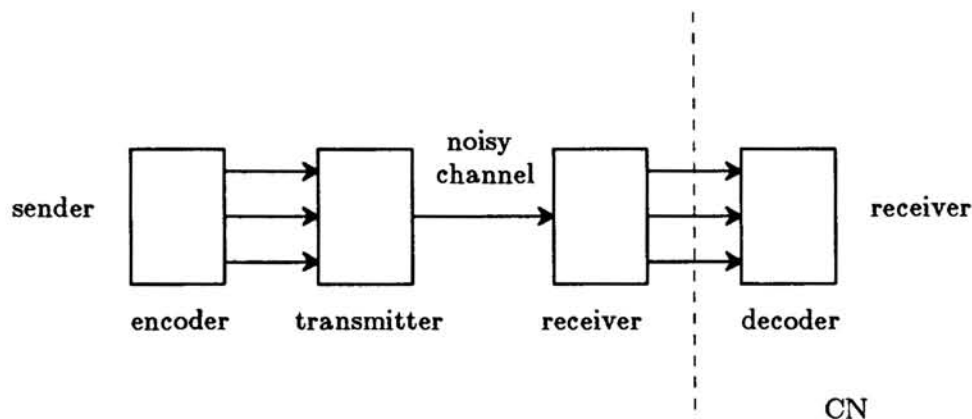

Figure 2 - A Transmission Channel

The operation of the channel is as follows. A bit is input into the channel encoder, which selects a random code of length $N$ and transmits that code over the channel. The receiver then, using nearest neighbor classification, decides if the original message was either a 0 or a 1.

Let $M$ be the number of code words used by the encoder. The rate* then indicates the density of the code space.

*Definition 7:* The *rate*, $R$, of a communication channel is

$$R \equiv \frac{\log M}{N} \tag{6}$$

□

The block length, $N$, corresponds directly to the receptive field size of the codon, i.e., $N = f_c$. The derivations in later sections use a related measure:

*Definition 8:* The *code utilization*, $b$, is the number of learned vectors assigned to a particular code or

$$b \equiv \frac{M}{2^N} \tag{7}$$

$b$ can be written in terms of R

$$b = 2^{N(R-1)} \tag{8}$$

As $b$ approaches 1, code compression increases. $b$ is essentially unbounded, since $M$ may be significantly larger than $2^N$. □

The decode error (information loss) due to code compression is a random variable that depends on the compression rate and the *a priori* probabilities, therefore, it will be different with different learned vector sets and codons within a set. As the average code utilization for all codons approaches 1, code compression occurs more often and codon decode error is unavoidable.

Let $\overline{x}_i$ be the vector output of the encoder, and the input to the channel, where each element of $\overline{x}_i$ is either a 1 or 0. Let $\overline{y}_i$ be the vector output of the channel, and the input to the decoder, where each element is either a 1 or a 0. The Noisy Channel Coding Theorem is now presented for a general case, where the individual $M$ input codes are to be distinguished. The result is then extended to a CN, where, even though $M$ input codes are used, the CN need only distinguish those codes where it must output a 1 from those where it must output a 0. The theorem is from Gallager (5.6.1)[5]. Random codes are assumed throughout.

*Theorem 1:* Let a discrete memoryless channel have transition probabilities $P_N(j/k)$ and, for any positive integer $N$ and positive number $R$, consider the ensemble of $(N,R)$ block codes in which each letter of each code word is independently selected according to the probability assignment $Q(k)$. Then, for each message $m$, $1 \leq m \leq \left\lceil e^{NR} \right\rceil$, and all $\rho$, $0 \leq \rho \leq 1$, the ensemble average probability of decoding error using maximum-likelyhood decoding satisfies

$$P'_{e,m} \leq \exp\left\{-N\left[E_0(\rho,Q) - \rho R\right]\right\} \tag{9}$$

where

$$E_0(\rho,Q)=-\ln\sum_{j=0}^{J-1}\left[\sum_{k=0}^{K-1}Q(k)P(j/k)^{\frac{1}{1+\rho}}\right]^{1+\rho} \tag{10}$$

□

These results are now adjusted for our special case.

*Theorem 2:* For a single CN, the average channel error rate for random code vectors is

$$P_{cdn}\leq 2q(1-q)P'_{e,m} \tag{11}$$

where $q=Q(k)\ \forall\ k$ is the probability of an input vector bit being a 1. □

These results cover a wide range of models. A more easily computable expression can be derived by recognizing some of the restrictions inherent in the CN model. First, assume that all channel code bits are equally likely, that is, $\forall\ k$, $Q(k)=q$, that the error model is the Binary Symmetric Channel (BSC), and that the errors are identically distributed and independent — that is, each bit has the same probability, $\epsilon$, of being in error, independent of the code word and the bit position in the code word.

A simplified version of the above theorem can be derived. Maximizing $\rho$ gives the tightest bounds:

$$P_{cdn}\leq 0.5\max_{0\leq\rho\leq 1}P'_e(\rho) \tag{12}$$

where (letting codon input be the block length, $N=f_c$)

$$P'_e(\rho)\leq\exp\left\{-f_c[E_0(\rho)-\rho R]\right\} \tag{13}$$

The minimum value of this expression is obtained when $\rho=1$ (for $q=0.5$):

$$E_0=-\log 2\left[\left(0.5\sqrt{\epsilon}+0.5\sqrt{1-\epsilon}\right)^2\right] \tag{14}$$

## SINGLE-CODON WITH CODE COMPRESSION

Unfortunately, the implementation complexity of a codon grows exponentially with the size of the codon, which limits its practical size. An alternative is to approximate single codon function of a single CN with many smaller, overlapped codons. The goal is to maintain performance and reduce implementation costs, thus improving the cost/performance of the decoding process. As codons get smaller, the receptive field size becomes smaller relative to the number of CNs in the network. When this happens there is codon compression, or *vector aliasing*, that introduces its own errors into the decoding process due to information loss. Networks can overcome this error by using multiple redundant codons (with overlapping receptive fields) that tend to correct the compression error.

Compression occurs when two code words requiring different decoder output share the same representation (within the receptive field of the codon). The following theorem gives the probability of incorrect codon output with and without compression error.

*Theorem 3:* For a BSC model where $q=0.5$, the codon receptive field is $f_c$, the code utilization is $b$, and the channel bits are selected randomly and independently, the probability of a codon decoding error when $b>1$ is approximately

$$P_{cdn}\leq(1-\epsilon)^{f_c}\overline{p}_c-\left[1-(1-\epsilon)^{f_c}\right]0.5 \tag{15}$$

where the expected compression error per codon is approximated by

$$\bar{p}_c = 0.5 - \frac{2\sqrt{bq(1-q)}}{b\sqrt{2\pi}} \tag{16}$$

and from equations 13-14, when $b < 1$

$$P_{cdn} \leq \exp\left\{-f_c\left[-\log\left(\left[(0.5\sqrt{\epsilon} + 0.5\sqrt{1-\epsilon})\right]^2\right) - R\right]\right\} \tag{17}$$

Proof is given in Hammerstrom[6] . □

As $b$ grows, $\bar{p}_c$ approaches 0.5 asymptotically. Thus, the performance of a single codon degrades rapidly in the presence of even small amounts of compression.

## MULTIPLE CODONS WITH CODE COMPRESSION

The use of multiple small codons is more efficient than a few large codons, but there are some fundamental performance constraints. When a codon is split into two or more smaller codons (and the original receptive field is subdivided accordingly), there are several effects to be considered. First, the error rate of each new codon increases due to a decrease in receptive field size (the codon's block code length). The second effect is that the code utilization, $b$, will increase for each codon, since the same number of learned vectors is mapped into a smaller receptive field. This change also increases the error rate per codon due to code compression. In fact, as the individual codon receptive fields get smaller, significant code compression occurs. For higher-order input codes, there is an added error that occurs when the order of the individual codons is decreased (since random codes are being assumed, this effect is not considered here). The third effect is the mass action of large numbers of codons. Even though individual codons may be in error, if the majority are correct, then the CN will have correct output. This effect decreases the total error rate.

Assume that each CN has more than one codon, $c > 1$. The union of the receptive fields for these codons is the receptive field for the CN with no no restrictions on the degree of overlap of the various codon receptive fields within or between CNs. For a CN with a large number of codons, the codon overlap will generally be random and uniformly distributed. Also assume that the transmission errors seen by different receptive fields are independent.

Now consider what happens to a codon's compression error rate (ignoring transmission error for the time being) when a codon is replaced by two or more smaller codons covering the same receptive field. This replacement process can continue until there are only 1-codons, which, incidentally, is analogous to most current neural models. For a multiple codon CN, assume that each codon votes a 1 or 0. The summation unit then totals this information and outputs a 1 if the majority of codons vote for a 1, etc.

*Theorem 4:* The probability of a CN error due to compression error is

$$P_c = \frac{1}{\sqrt{2\pi}} \int_{\frac{c/2 - c\bar{p}_c - 1/2}{\sqrt{c\bar{p}_c(1-\bar{p}_c)}}}^{\infty} e^{-\frac{1}{2}x^2} dy \tag{18}$$

where $\bar{p}_c$ is given in equation 16 and $q = 0.5$.

$P_c$ incorporates the two effects of moving to multiple smaller codons and adding more codons. Using equation 17 gives the total error probability (per bit), $P_{CN}$:

$$P_{CN} = P_{cdn} + P_c - P_{cdn}P_c \tag{19}$$

Proof is in Hammerstrom[6] . □

For networks that perform association as defined in this paper, the connection weights rapidly approach a single uniform value as the size of the network grows. In information theoretic terms, the information content of those weights approaches zero as the compression increases. Why then do simple non-conjunctive networks (1-codon equivalent) work at all? In the next section I define connectivity cost constraints and show that the answer to the first question is that the general associative structures defined here *do not* scale cost-effectively and more importantly that there are limits to the degree of distribution of information.

## CONNECTIVITY COSTS

It is much easier to assess costs if some implementation medium is assumed. I have chosen standard silicon, which is a two dimensional surface where CN's and codons take up surface area according to their receptive field sizes. In addition, there is area devoted to the metal lines that interconnect the CNs. A specific VLSI technology need not be assumed, since the comparisons are relative, thus keeping CNs, codons, and metal in the proper proportions, according to a standard metal width, $m_w$ (which also includes the inter-metal pitch). For the analyses performed here, it is assumed that $m_l$ levels of metal are possible.

In the previous section I established the relationship of network performance, in terms of the transmission error rate, $\epsilon$, and the network capacity, $M$. In this section I present an implementation cost, which is total silicon area, $A$. This figure can then be used to derive a cost/performance figure that can be used to compare such factors as codon size and receptive field size. There are two components to the total area: $A_{CN}$, the area of a CN, and $A_{MI}$, the area of the metal interconnect between CNs. $A_{CN}$ consists of the silicon area requirements of the codons for all CNs. The metal area for local, intra-CN interconnect is considered to be much smaller than that of the codons themselves and of that of the more global, inter-CN interconnect, and is not considered here. The area per CN is roughly

$$A_{CN} = c f_c m_c \left(\frac{m_w}{m_l}\right)^2 \qquad (20)$$

where $m_c$ is the maximum number of vectors that each codon must distinguish, for $b \geq 1$, $m_c = 2^{f_c}$.

*Theorem 5:* Assume a rectangular, *unbounded** grid of CNs (all CNs are equi-distant from their four nearest neighbors), where each CN has a bounded receptive field of its $n_{CN}$ nearest CNs, where $n_{CN}$ is the receptive field size for the CN, $n_{CN} = \dfrac{c f_c}{R}$, where $c$ is the number of codons, and $R$ is the intra-CN redundancy, that is, the ratio of inputs to synapses (e.g., when $R=1$ each CN input is used once at the CN, when $R=2$ each input is used on the average at two sites). The metal area required to support each CN's receptive field is (proof is giving by Hammerstrom[6] ):

$$A_{MI} = \left[\frac{n_{CN}^3}{16} + \frac{3 n_{CN}^{\frac{5}{2}}}{2} + 9 n_{CN}^2\right] \left(\frac{m_w}{m_l}\right)^2 \qquad (21)$$

The total area per CN, $A$, then is

---

*Another implementation strategy is to place all CNs along a diagonal, which gives $n^2$ area. However, this technique only works for a *bounded* number of CNs and when dendritic computation can be spread over a large area, which limits the range of possible CN implementations. The theorem stated here covers an infinite plane of CNs each with a *bounded* receptive field.

$$A = (A_{MI} + A_{CN}) = O(n_{CN}^3) \qquad (22)$$

☐

Even with the assumption of maximum locality, the total metal interconnect area increases as the *cube* of the per CN receptive field size!

## SINGLE CN SIMULATION

What do the bounds tell us about CN connectivity requirements? From simulations, increasing the CN's receptive field size improves the performance (increases capacity), but there is also an increasing cost, which increases faster than the performance! Another observation is that redundancy is quite effective as a means for increasing the effectiveness of a CN with constrained connectivity. (There are some limits to $R$, since it can reach a point where the intra-CN connectivity approaches that of inter-CN for some situations.) With a fixed $n_{CN}$, increasing cost-effectiveness $(A/m)$ is possible by increasing both order and redundancy.

In order to verify the derived bounds, I also wrote a discrete event simulation of a CN, where a random set of learned vectors were chosen and the CN's codons were programmed according to the model presented earlier. Learned vectors were chosen randomly and subjected to random noise, $\epsilon$. The CN then attempted to categorize these inputs into two major groups (CN output = 1 and CN output = 0). For the most part the analytic bounds agreed with the simulation, though they tended to be optimistic in slightly underestimating the error. These differences can be easily explained by the simplifying assumptions that were made to make the analytic bounds mathematically tractable.

## DISTRIBUTED VS. LOCALIZED

Throughout this paper, it has been tacitly assumed that representations are distributed across a number of CNs, and that any single CN participates in a number of representations. In a *local* representation each CN represents a single concept or feature. It is the distribution of representation that makes the CN's decode job difficult, since it is the cause of the code compression problem.

There has been much debate in the connectionist/neuromodelling community as to the advantages and disadvantages of each approach; the interested reader is referred to Hinton[7], Baum et al. [8], and Ballard[9]. Some of the results derived here are relevant to this debate. As the distribution of representation increases, the compression per CN increases accordingly. It was shown above that the mean error in a codon's response quickly approaches 0.5, independent of the input noise. This result also holds at the CN level. For each individual CN, this error can be offset by adding more codons, but this is expensive and tends to obviate one of the arguments in favor of distributed representations, that is, the multi-use advantage, where fewer CNs are needed because of more complex, redundant encodings. As the degree of distribution increases, the required connectivity and the code compression increases, so the added information that each codon adds to its CN's decoding process goes to zero (equivalent to all weights approaching a uniform value).

## SUMMARY AND CONCLUSIONS

In this paper a single CN (node) performance model was developed that was based on Communication Theory. Likewise, an implementation cost model was derived.

The communication model introduced the codon as a higher-order decoding element and showed that for small codons (much less than total CN fan-in, or convergence) code compression, or vector aliasing, within the codon's receptive field is a severe problem for

large networks. As code compression increases, the information added by any individual codon to the CN's decoding task rapidly approaches zero.

The cost model showed that for 2-dimensional silicon, the area required for inter-node metal connectivity grows as the cube of a CN's fan-in.

The combination of these two trends indicates that past a certain point, which is highly dependent on the probability structure of the learned vector space, increasing the fan-in of a CN (as is done, for example, when the distribution of representation is increased) yields diminishing returns in terms of total cost-performance. Though the rate of diminishing returns can be decreased by the use of redundant, higher-order connections.

The next step is to apply these techniques to ensembles of nodes (CNs) operating in a competitive learning or feature extraction environment.

## Footnotes

*In the definitions given here and the theorems below, the notation of Gallager[5] is used. Many of the definitions and theorems are also from Gallager.

## REFERENCES

[1]     J. Bailey, "A VLSI Interconnect Structure for Neural Networks," Ph.D. Dissertation, Department of Computer Science/Engineering, OGC. In Preparation.

[2]     V. B. Mountcastle, "An Organizing Principle for Cerebral Function: The Unit Module and the Distributed System," in *The Mindful Brain*, MIT Press, Cambridge, MA, 1977.

[3]     T. Maxwell, C. L. Giles, Y. C. Lee and H. H. Chen, "Transformation Invariance Using High Order Correlations in Neural Net Architectures," *Proceedings International Conf. on Systems, Man, and Cybernetics*, 1986.

[4]     D. Marr, "A Theory for Cerebral Neocortex," *Proc. Roy. Soc. London*, vol. 176(1970), pp. 161-234.

[5]     R. G. Gallager, *Information Theory and Reliable Communication*, John Wiley and Sons, New York, 1968.

[6]     D. Hammerstrom, "A Connectivity Analysis of Recursive, Auto-Associative Connection Networks," Tech. Report CS/E-86-009, Dept. of Computer Science/Engineering, Oregon Graduate Center, Beaverton, Oregon, August 1986.

[7]     G. E. Hinton, "Distributed Representations," Technical Report CMU-CS-84-157, Computer Science Dept., Carnegie-Mellon University, Pittsburgh, PA 15213, 1984.

[8]     E. B. Baum, J. Moody and F. Wilczek, "Internal Representations for Associative Memory," Technical Report NSF-ITP-86-138, Institute for Theoretical Physics, Santa Barbara, CA, 1986.

[9]     D. H. Ballard, "Cortical Connections and Parallel Processing: Structure and Function," Technical Report 133, Computer Science Department, Rochester, NY, January 1985.